# Tighter Bounds for Structured Estimation

**Chuong B. Do, Quoc Le**
Stanford University
{chuongdo,quocle}@cs.stanford.edu

**Choon Hui Teo**
Australian National University and NICTA
choonhui.teo@anu.edu.au

**Olivier Chapelle, Alex Smola**
Yahoo! Research
chap@yahoo-inc.com,alex@smola.org

## Abstract

Large-margin structured estimation methods minimize a convex upper bound of loss functions. While they allow for efficient optimization algorithms, these convex formulations are not tight and sacrifice the ability to accurately model the true loss. We present tighter non-convex bounds based on generalizing the notion of a ramp loss from binary classification to structured estimation. We show that a small modification of existing optimization algorithms suffices to solve this modified problem. On structured prediction tasks such as protein sequence alignment and web page ranking, our algorithm leads to improved accuracy.

## 1 Introduction

Structured estimation [18, 20] and related techniques has proven very successful in many areas ranging from collaborative filtering to optimal path planning, sequence alignment, graph matching and named entity tagging.

At the heart of those methods is an inverse optimization problem, namely that of finding a function $f(x,y)$ such that the prediction $y^*$ which maximizes $f(x,y^*)$ for a given $x$, minimizes some loss $\Delta(y,y^*)$ on a training set. Typically $x \in \mathfrak{X}$ is referred to as a pattern, whereas $y \in \mathcal{Y}$ is a corresponding label. $\mathcal{Y}$ can represent a rich class of possible data structures, ranging from binary sequences (tagging), to permutations (matching and ranking), to alignments (sequence matching), to path plans [15]. To make such inherently discontinuous and nonconvex optimization problems tractable, one applies a convex upper bound on the incurred loss. This has two benefits: firstly, the problem has no local minima, and secondly, the optimization problem is continuous and piecewise differentiable, which allows for effective optimization [17, 19, 20]. This setting, however, exhibits a significant problem: the looseness of the convex upper bounds can sometimes lead to poor accuracy.

For binary classification, [2] proposed to switch from the hinge loss, a convex upper bound, to a tighter nonconvex upper bound, namely the ramp loss. Their motivation was not the accuracy though, but the faster optimization due to the decreased number of support vectors. The resulting optimization uses the convex-concave procedure of [22], which is well known in optimization as the DC-programming method [9].

We extend the notion of ramp loss to structured estimation. We show that with some minor modifications, the DC algorithms used in the binary case carry over to the structured setting. Unlike the binary case, however, we observe that for structured prediction problems with noisy data, DC programming can lead to improved accuracy in practice. This is due to increased robustness. Effectively, the algorithm discards observations which it labels incorrectly if the error is too large. This ensures that one ends up with a lower-complexity solution while ensuring that the "correctable" errors are taken care of.

## 2 Structured Estimation

Denote by $\mathcal{X}$ the set of patterns and let $\mathcal{Y}$ be the set of labels. We will denote by $X := \{x_1, \ldots, x_m\}$ the observations and by $Y := \{y_1, \ldots, y_m\}$ the corresponding set of labels. Here the pairs $(x_i, y_i)$ are assumed to be drawn from some distribution $\Pr$ on $\mathcal{X} \times \mathcal{Y}$.

Let $f : \mathcal{X} \times \mathcal{Y} \to \mathbb{R}$ be a function defined on the product space. Finally, denote by $\Delta : \mathcal{Y} \times \mathcal{Y} \to \mathbb{R}_0^+$ a loss function which maps pairs of labels to nonnegative numbers. This could be, for instance, the number of bits in which $y$ and $y'$ differ, i.e. $\Delta(y, y') = \|y - y'\|_1$ or considerably more complicated loss functions, e.g., for ranking and retrieval [21]. We want to find $f$ such that for

$$y^*(x, f) := \underset{y'}{\operatorname{argmax}} f(x, y') \tag{1}$$

the loss $\Delta(y, y^*(x, f))$ is minimized: given $X$ and $Y$ we want to minimize the regularized risk,

$$R_{\mathrm{reg}}[f, X, Y] := \frac{1}{m} \sum_{i=1}^{m} \Delta(y_i, y^*(x_i, f)) + \lambda \Omega[f]. \tag{2}$$

Here $\Omega[f]$ is a regularizer, such as an RKHS norm $\Omega[f] = \|f\|_{\mathcal{H}}^2$ and $\lambda > 0$ is the associated regularization constant, which safeguards us against overfitting. Since (2) is notoriously hard to minimize several convex upper bounds have been proposed to make $\Delta(y_i, y^*(x_i, f))$ tractable in $f$. The following lemma, which is a generalization of a result of [20] provides a strategy for convexification:

**Lemma 1** *Denote by* $\Gamma : \mathbb{R}_0^+ \to \mathbb{R}_0^+$ *a monotonically increasing nonnegative function. Then*

$$l(x, y, y'', f) := \sup_{y'} \Gamma(\Delta(y, y')) \left[ f(x, y') - f(x, y'') \right] + \Delta(y, y') \geq \Delta(y, y^*(x, f))$$

*for all* $y, y'' \in \mathcal{Y}$. *Moreover,* $l(x, y, y'', f)$ *is convex in* $f$.

**Proof** Convexity follows immediately from the fact that $l$ is the supremum over linear functions in $f$. To see the inequality, plug $y' = y^*(x, f)$ into the LHS of the inequality: by construction $f(x, y^*(x, f)) \geq f(x, y'')$ for all $y'' \in \mathcal{Y}$. ∎

In regular convex structured estimation, $l(x, y, y, f)$ is used. Methods in [18] choose the constant function $\Gamma(\eta) = 1$, whereas methods in [20] choose margin rescaling by means of $\Gamma(\eta) = \eta$. This also shows why both formulations lead to convex upper bounds of the loss. It depends very much on the form of $f$ and $\Delta$ which choice of $\Gamma$ is easier to handle. Note that the inequality holds for all $y''$ rather than only for the "correct" label $y'' = y$. We will exploit this later.

## 3 A Tighter Bound

For convenience denote by $\beta(x, y, y', f)$ the relative margin between $y$ and $y'$ induced by $f$ via

$$\beta(x, y, y', f) := \Gamma(\Delta(y, y'))[f(x, y') - f(x, y)]. \tag{3}$$

The loss bound of Lemma 1 suffers from a significant problem: for large values of $f$ the loss may grow without bound, provided that the estimate is incorrect. This is not desirable since in this setting even a single observation may completely ruin the quality of the convex upper bound on the misclassification error.

Another case where the convex upper bound is not desirable is the following: imagine that there are a lot of $y$ which are as good as the label in the training set; this happens frequently in ranking where there are ties between the optimal permutations. Let us denote by $\mathcal{Y}_{opt} := \{y''$ such that $\Delta(y, y') = \Delta(y'', y'), \forall y'\}$ this set of equally good labels. Then one can replace $y$ by any element of $\mathcal{Y}_{opt}$ in the bound of Lemma 1. Minimization over $y'' \in \mathcal{Y}_{opt}$ leads to a tighter non-convex upper bound:

$$l(x, y, y, f) \geq \inf_{y'' \in \mathcal{Y}_{opt}} \sup_{y'} \beta(x, y'', y', f) + \Delta(y'', y') \geq \Delta(y, y^*(x, f)).$$

In the case of binary classification, [2] proposed the following non-convex loss that can be minimized using DC programming:

$$l(x, y, f) := \min(1, \max(0, 1 - yf(x))) = \max(0, 1 - yf(x)) - \max(0, -yf(x)). \tag{4}$$

We see that (4) is the difference between a soft-margin loss and a hinge loss. That is, the difference between a loss using a *large* margin related quantity and one using simply the violation of the margin. This difference ensures that $l$ cannot increase without bound, since in the limit the derivative of $l$ with respect to $f$ vanishes. The intuition for extending this to structured losses is that the generalized hinge loss underestimates the actual loss whereas the soft margin loss overestimates the actual loss. Taking the difference removes linear scaling behavior while retaining the continuous properties.

**Lemma 2** *Denote as follows the rescaled estimate and the margin violator*

$$\tilde{y}(x, y, f) := \operatorname*{argmax}_{y'} \beta(x, y, y', f) \ and \ \bar{y}(x, y, f) := \operatorname*{argmax}_{y'} \beta(x, y, y', f) + \Delta(y, y') \quad (5)$$

*Moreover, denote by $l(x, y, f)$ the following loss function*

$$l(x, y, f) := \sup_{y'}[\beta(x, y, y', f) + \Delta(y, y')] - \sup_{y'} \beta(x, y, y', f). \quad (6)$$

*Then under the assumptions of Lemma 1 the following bound holds*

$$\Delta(y, \bar{y}(x, y, f)) \geq l(x, y, f) \geq \Delta(y, y^*(x, f)) \quad (7)$$

This loss is a difference between two convex functions, hence it may be (approximately) minimized by a DC programming procedure. Moreover, it is easy to see that for $\Gamma(\eta) = 1$ and $f(x, y) = \frac{1}{2}yf(x)$ and $y \in \{\pm 1\}$ we recover the ramp loss of (4).

**Proof** Since $\bar{y}(x, y, f)$ maximizes the first term in (6), replacing $y'$ by $\bar{y}(x, y, f)$ in both terms yields

$$l(x, y, f) \leq \beta(x, y, \bar{y}, f) + \Delta(y, \bar{y}) - \beta(x, y, \bar{y}, f) = \Delta(y, \bar{y}).$$

To show the lower bound, we distinguish the following two cases:

**Case 1: $y^*$ is a maximizer of** $\sup_{y'} \beta(x, y, y', f)$
Replacing $y'$ by $y^*$ in both terms of (6) leads to $l(x, y, f) \geq \Delta(y, y^*)$.
**Case 2: $y^*$ is not a maximizer of** $\sup_{y'} \beta(x, y, y', f)$
Let $\tilde{y}$ be any maximizer. Because $f(x, y^*) \geq f(x, \tilde{y})$, we have $\Gamma(\Delta(y, \tilde{y})) [f(x, y^*) - f(x, y)] > \Gamma(\Delta(y, \tilde{y})) [f(x, \tilde{y}) - f(x, y)] > \Gamma(\Delta(y, y^*)) [f(x, y^*) - f(x, y)]$ and thus $\Gamma(\Delta(y, \tilde{y})) > \Gamma(\Delta(y, y^*))$. Since $\Gamma$ is non-decreasing this implies $\Delta(y, \tilde{y}) > \Delta(y, y^*)$. On the other hand, plugging $\tilde{y}$ in (6) gives $l(x, y, f) \geq \Delta(y, \tilde{y})$. Combining both inequalities proves the claim. ∎

Note that the main difference between the cases of constant $\Gamma$ and monotonic $\Gamma$ is that in the latter case the bounds are not quite as tight as they could potentially be, since we still have some slack with respect to $\Delta(y, \tilde{y})$. Monotonic $\Gamma$ tend to overscale the margin such that more emphasis is placed on avoiding large deviations from the correct estimate rather than restricting small deviations.

Note that this nonconvex upper bound is not likely to be Bayes consistent. However, it will generate solutions which have a smaller model complexity since it is never larger than the convex upper bound on the loss, hence the regularizer on $f$ plays a more important role in regularized risk minimization. As a consequence one can expect better statistical concentration properties.

## 4 DC Programming

We briefly review the basic template of DC programming, as described in [22]. For a function

$$f(x) = f_{\mathrm{cave}}(x) + f_{\mathrm{vex}}(x)$$

which can be expressed as the sum of a convex $f_{\mathrm{vex}}$ and a concave $f_{\mathrm{cave}}$ function, we can find a convex *upper* bound by $f_{\mathrm{cave}}(x_0) + \langle x - x_0, f'_{\mathrm{cave}}(x_0)\rangle + f_{\mathrm{vex}}(x)$. This follows from the first-order Taylor expansion of the concave part $f_{\mathrm{cave}}$ at the current value of $x$. Subsequently, this upper bound is minimized, a new Taylor approximation is computed, and the procedure is repeated. This will lead to a local minimum, as shown in [22].

We now proceed to deriving an explicit instantiation for structured estimation. To keep things simple, in particular the representation of the functional subgradients of $l(x, y, f)$ with respect to $f$, we assume that $f$ is drawn from a Reproducing Kernel Hilbert Space $\mathcal{H}$.

---
**Algorithm 1** Structured Estimation with Tighter Bounds

---
Using the loss of Lemma 1 initialize $f = \operatorname{argmin}_{f'} \ \sum_{i=1}^{m} l(x_i, y_i, y_i, f') + \lambda\Omega[f']$
**repeat**
    Compute $\tilde{y}_i := \tilde{y}(x_i, y_i, f)$ for all $i$.
    Using the tightened loss bound recompute $f = \operatorname{argmin}_{f'} \sum_{i=1}^{m} \tilde{l}(x_i, y_i, \tilde{y}_i, f') + \lambda\Omega[f']$
**until** converged

---

Denote by $k$ the kernel associated with $\mathcal{H}$, defined on $(\mathcal{X} \times \mathcal{Y}) \times (\mathcal{X} \times \mathcal{Y})$. In this case for $f \in \mathcal{H}$ we have by the reproducing property that $f(x, y) = \langle f, k((x, y), \cdot)\rangle$ and the functional derivative is given by $\partial_f f(x, y) = k((x, y), \cdot)$. Likewise we may perform the linearization in (6) as follows:

$$-\sup_{y'} \beta(x, y, y', f) \leq -\beta(x, y, \tilde{y}, f)$$

In other words, we use the rescaled estimate $\tilde{y}$ to provide an upper bound on the concave part of the loss function. This leads to the following instantiation of standard convex-concave procedure: instead of the structured estimation loss it uses the loss bound $\tilde{l}(x, y, \tilde{y}, f)$

$$\tilde{l}(x, y, \tilde{y}, f) := \sup_{y' \in \mathcal{Y}} [\beta(x, y, y', f) + \Delta(y, y')] - \beta(x, y, \tilde{y}, f)$$

In the case of $\Gamma(\eta) = 1$ this can be simplified significantly: the terms in $f(x, y)$ cancel and $\tilde{l}$ becomes

$$\tilde{l}(x, y, \tilde{y}, f) = \sup_{y' \in \mathcal{Y}} [f(x, y') - f(x, \tilde{y})] + \Delta(y, y').$$

In other words, we replace the correct label $y$ by the rescaled estimate $\tilde{y}$. Such modifications can be easily implemented in bundle method solvers and related algorithms which only require access to the gradient information (and the function value). In fact, the above strategy follows directly from Lemma 1 when replacing $y''$ by the rescaled estimate $\tilde{y}$.

## 5 Experiments

### 5.1 Multiclass Classification

In this experiment, we investigate the performance of convex and ramp loss versions of the Winner-Takes-All multiclass classification [1] when the training data is *noisy*. We performed the experiments on some UCI/Statlog datasets: DNA, LETTER, SATIMAGE, SEGMENT, SHUTTLE, and USPS, with some fixed percentages of the labels shuffled, respectively. Note that we reshuffled the labels in a stratified fashion. That is, we chose a fixed fraction from each class and we permuted the label assignment subsequently.

Table 1 shows the results (average accuracy $\pm$ standard deviation) on several datasets with different percentages of labels shuffled. We used nested 10-fold crossvalidation to adjust the regularization constant and to compute the accuracy. A linear kernel was used. It can be seen that ramp loss outperforms the convex upper bound when the datasets are noisy. For clean data the convex upper bound is slightly superior, albeit not in a statistically significant fashion. This supports our conjecture that, compared to the convex upper bound, the ramp loss is more robust on noisy datasets.

### 5.2 Ranking with Normalized Discounted Cumulative Gains

Recently, [12] proposed a method for learning to rank for web search. They compared several methods showing that optimizing the Normalized Discounted Cumulative Gains (NDCG) score using a form of structured estimation yields best performance. The algorithm used a linear assignment problem to deal with ranking.

In this experiment, we perform ranking experiments with the OHSUMED dataset which is publicly available [13]. The dataset is already preprocessed and split into 5 folds. We first carried out the structured output training algorithm which optimizes the convex upper bound of NDCG as described in [21]. Unfortunately, the returned solution was $f = 0$. The convex upper bounds led to the

| Dataset | Methods | 0% | 10% | 20% |
|---------|---------|-----|------|------|
| DNA | convex | $95.2 \pm 1.1$ | $88.9 \pm 1.5$ | $83.1 \pm 2.4$ |
|  | ramp loss | $95.1 \pm 0.8$ | $89.1 \pm 1.3$ | $83.5 \pm 2.2$ |
| LETTER | convex | $76.8 \pm 0.9$ | $64.6 \pm 0.7$ | $50.1 \pm 1.4$ |
|  | ramp loss | $78.6 \pm 0.8$ | $70.8 \pm 0.8$ | $63.0 \pm 1.5$ |
| SATIMAGE | convex | $85.1 \pm 0.9$ | $77.0 \pm 1.6$ | $66.4 \pm 1.3$ |
|  | ramp loss | $85.4 \pm 1.2$ | $78.1 \pm 1.6$ | $70.7 \pm 1.0$ |
| SEGMENT | convex | $95.4 \pm 0.9$ | $84.8 \pm 2.3$ | $73.8 \pm 2.1$ |
|  | ramp loss | $95.2 \pm 1.0$ | $85.9 \pm 2.1$ | $77.5 \pm 2.0$ |
| SHUTTLE | convex | $97.4 \pm 0.2$ | $89.5 \pm 0.2$ | $83.8 \pm 0.2$ |
|  | ramp loss | $97.1 \pm 0.2$ | $90.6 \pm 0.8$ | $88.1 \pm 0.3$ |
| USPS | convex | $95.1 \pm 0.7$ | $85.3 \pm 1.3$ | $76.5 \pm 1.4$ |
|  | ramp loss | $95.1 \pm 0.9$ | $86.1 \pm 1.6$ | $77.6 \pm 1.1$ |

Table 1: Average accuracy for multiclass classification using the convex upper bound and the ramp loss. The third through fifth columns represent results for datasets with none, 10%, and 20% of the labels randomly shuffled, respectively.

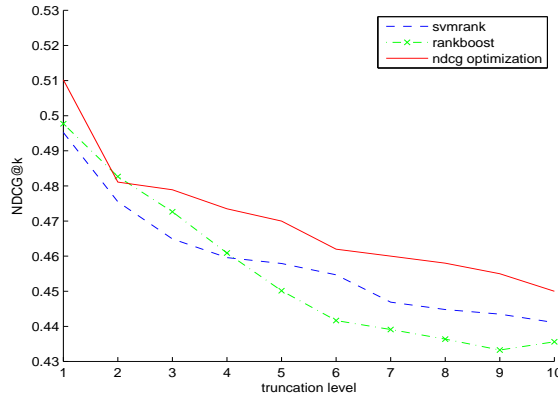

Figure 1: NDCG comparison against ranking SVM and RankBoost. We report the NDCG computed at various truncation levels. Our non-convex upper bound consistently outperforms other rankers. In the context of web page ranking an improvement of $0.01 - 0.02$ in the NDCG score is considered substantial.

undesirable situation where no nonzero solution would yield any improvement, since the linear function class was too simple.

This problem is related to the fact that there are a lot of rankings which are equally good because of the ties in the editorial judgments (see beginning of section 3). As a result, there is no $w$ that learns the data well, and for each $w$ the associated $\max_{y'} f(x, y') - f(x, y) + \Delta(y, y')$ causes either the first part or the second part of the loss to be big such that the total value of the loss function always exceeds $\max \Delta(y, y')$.

When using the non-convex formulation the problem can be resolved because we do not entirely rely on the $y$ given in the training set, but instead find the $y$ that minimizes the loss. We compared the results of our method and two standard methods for ranking: ranking SVM [10, 8] and RankBoost [6] (the baselines for OHSUMED are shown in [13]) and used NDCG as the performance criterion. We report the aggregate performance in Figure 1.

As can be seen from the figure, the results from the new formulation are better than standard methods for ranking. It is worth emphasizing that the most important contribution is not only that the new formulation can give comparable results to the state-of-the-art algorithms for ranking but also that it provides *useful* solutions when the convex structured estimation setting provides only useless results (obviously $f = 0$ is highly undesirable).

## 5.3 Structured classification

We also assessed the performance of the algorithm on two different structured classification tasks for computational biology, namely protein sequence alignment and RNA secondary structure prediction.

**Protein sequence alignment** is the problem of comparing the amino acid sequences corresponding to two different proteins in order to identify regions of the sequences which have common ancestry or biological function. In the pairwise sequence alignment task, the elements of the input space $\mathcal{X}$ consist of pairs of amino acid sequences, represented as strings of approximately 100-1000 char-

| Method | 0-10% (324) | 11-20% (793) | 21-30% (429) | 31-40% (239) | Overall (1785) |
|---|---|---|---|---|---|
| CRF | 0.111 | 0.316 | 0.634 | 0.877 | 0.430 |
| convex | 0.116 | 0.369 | 0.699 | 0.891 | 0.472 |
| ramp loss | 0.138 | 0.387 | 0.708 | 0.905 | 0.488 |

Table 2: Protein pairwise sequence alignment results, stratified by reference alignment percentage identity. The second through fifth columns refer to the four non-overlapping reference alignment percentage identity ranges described in the text, and the sixth column corresponds to overall results, pooled across all four subsets. Each non-bolded value represents the average test set recall for a particular algorithm on alignment from the corresponding subset. The numbers in parentheses indicate the total number of sequences in each subset.

| Method | 1-50 (118) | 51-100 (489) | 101-200 (478) | 201+ (274) | Overall (1359) |
|---|---|---|---|---|---|
| CRF | 0.546 / 0.862 | 0.586 / 0.727 | 0.467 / 0.523 | 0.414 / 0.472 | 0.505 / 0.614 |
| convex | 0.690 / 0.755 | 0.664 / 0.629 | 0.571 / 0.501 | 0.542 / 0.484 | 0.608 / 0.565 |
| ramp loss | 0.725 / 0.708 | 0.705 / 0.602 | 0.612 / 0.489 | 0.569 / 0.461 | 0.646 / 0.542 |

Table 3: RNA secondary structure prediction results. The second through fifth columns represent subsets of the data stratified by sequence length. The last column presents overall results, pooled across all four subsets. Each pair of non-bolded numbers indicates the sensitivity / selectivity for structures in the two-fold cross-validation. The numbers in parentheses indicate the total number of sequences in each subset.

acters in length. The output space $\mathcal{Y}$ contains candidate alignments which identify the corresponding positions in the two sequences which are hypothesized to be evolutionarily related.

We developed a structured prediction model for pairwise protein sequence alignment, using the types of features described in [3, 11] For the loss function, we used $\Delta(y, y') = 1 - recall$ (where $recall$ is the proportion of aligned amino acid matches in the true alignment $y$ that appear in the predicted alignment $y'$. For each inner optimization step, we used a fast-converging subgradient-based optimization algorithm with an adaptive Polyak-like step size [23].

We performed two-fold cross-validation over a collection of 1785 pairs of structurally aligned protein domains [14]. All hyperparameters were selected via holdout cross validation on the training set, and we pooled the results from the two folds. For evaluation, we used recall, as described previously, and compared the performance of our algorithm to a standard conditional random field (CRF) model and max-margin model using the same features. The *percentage identity* of a reference alignment is defined as the proportion of aligned residue pairs corresponding to identical amino acids. We partitioned the alignments in the testing collection into four subsets based on percent identity (0-10%, 11-20%, 21-30%, and 31+%), showed the recall of the algorithm for each subset in addition to overall recall (see Table 2).

Here, it is clear that our method obtains better accuracy than both the CRF and max-margin models.[1] We note that the accuracy differences are most pronounced at the low percentage identity ranges, the 'twilight zone' regime where better alignment accuracy has far reaching consequences in many other computational biology applications [16].

**RNA secondary structure prediction** Ribonucleic acid (RNA) refers to a class of long linear polymers composed of four different types of nucleotides (A, C, G, U). Nucleotides within a single RNA molecule base-pair with each other, giving rise to a pattern of base-pairing known as the RNA's secondary structure. In the RNA secondary structure prediction problem, we are given an RNA sequence (a string of approximately 20-500 characters) and are asked to predict the secondary structure that the RNA molecule will form *in vivo*. Conceptually, an RNA secondary structure can be thought of as a set of unordered pairs of nucleotide indices, where each pair designates two

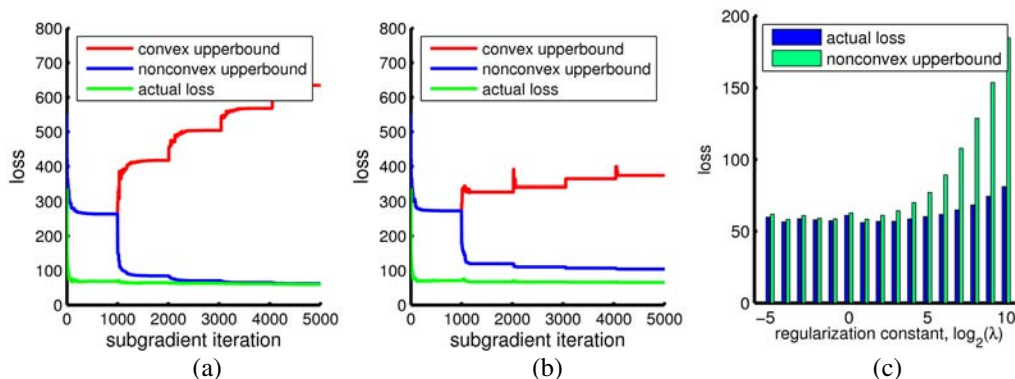

Figure 2: Tightness of the nonconvex bound. Figures (a) and (b) show the value of the nonconvex loss, the convex loss and the actual loss as a function of the number of iterations when minimizing the *nonconvex* upper bound. At each relinearization, which occurs every 1000 iterations, the nonconvex upper bound decreases. Note that the convex upper bound increases in the process as convex and nonconvex bound diverge further from each other. We chose $\lambda = 2^{-6}$ in Figure (a) and $\lambda = 2^7$ for Figure (b). Figure (c) shows the tightness of the final nonconvex bound at the end of optimization for different values of the regularization parameter $\lambda$.

nucleotides in the RNA molecule which base-pair with each other. Following convention, we take the structured output space $\mathcal{Y}$ to be the set of all possible pseudoknot-free structures.

We used a max-margin model for secondary structure prediction. The features of the model were chosen to match the energetic terms in standard thermodynamic models for RNA folding [4]. As our loss function, we used $\Delta(y, y') = 1 - recall$ (where $recall$ is the proportion of base-pairs in the reference structure $y$ that are recovered in the predicted structure $y'$). We again used the subgradient algorithm for optimization.

To test the algorithm, we performed two-fold cross-validation over a large collection of 1359 RNA sequences with known secondary structures from the RFAM database (release 8.1) [7]. We evaluated the methods using two standard metrics for RNA secondary structure prediction accuracy known as sensitivity and selectivity (which are the equivalent of recall and precision, respectively, for this domain). For reporting, we binned the sequences in the test collection by length into four ranges (1-50, 51-100, 101-200, 201+ nucleotides), and evaluated the sensitivity and selectivity of the algorithm for each subset in addition to overall accuracy (see Table 3).

Again, our algorithm consistently outperforms an equivalently parameterized CRF and max-margin model in terms of sensitivity.[2] The selectivity of the predictions from our algorithm is often worse than that of the other two models. This is likely because we opted for a loss function that penalizes for "false negative" base-pairings but not "false-positives" since our main interest is in identifying correct base-pairings (a harder task than predicting only a small number of high-confidence base-pairings). An alternative loss function that chooses a different balance between penalizing false positives and false negatives would achieve a different trade-off of sensitivity and selectivity.

**Tightness of the bound:** We generated plots of the convex, nonconvex, and actual losses (which correspond to $l(x, y, y, f)$, $l(x, y, f)$, and $\Delta(y, y^*(x, f))$, respectively, from Lemma 2) over the course of optimization for our RNA folding task (see Figure 2). From Figures 2a and 2b, we see that the nonconvex loss provides a much tighter upper bound on the actual loss function. Figure 2c shows that the tightness of the bound decreases for increasing regularization parameters $\lambda$.

In summary, our bound leads to improvements whenever there is a large number of instances $(x, y)$ which cannot be classified perfectly. This is not surprising as for "clean" datasets even the convex upper bound vanishes when no margin errors are encountered. Hence noticeable improvements can be gained mainly in the structured output setting rather than in binary classification.

## 6 Summary and Discussion

We proposed a simple modification of the convex upper bound of the loss in structured estimation which can be used to obtain tighter bounds on sophisticated loss functions. The advantage of our approach is that it requires next to no modification of existing optimization algorithms but rather repeated invocation of a structured estimation solver such as SVMStruct, BMRM, or Pegasos.

In several applications our approach outperforms the convex upper bounds. This can be seen both for multiclass classification, for ranking where we encountered underfitting and undesirable trivial solutions for the convex upper bound, and in the context of sequence alignment where in particular for the hard-to-align observations significant gains can be found.

From this experimental study, it seems that the tighter non-convex upper bound is useful in two scenarios: when the labels are noisy and when for each example there is a large set of labels which are (almost) as good as the label in the training set. Future work includes studying other types of structured estimation problems such as the ones encountered in NLP to check if our new upper bound can also be useful for these problems.

## Footnotes

[1]We note that the results here are based on using the Viterbi algorithm for parsing, which differs from the inference method used in [3]. In practice this is preferable to posterior decoding as it is significantly faster which is crucial applications to large amounts of data.

[2]Note that the results here are based on using the CYK algorithm for parsing, which differs from the inference method used in [4].

## References

[1] K. Crammer, and Y. Singer. On the Learnability and Design of Output Codes for Multiclass Problems. In *COLT 2000*, pages 35–46. Morgan Kaufmann, 2000.

[2] R. Collobert, F.H. Sinz, J. Weston, and L. Bottou. Trading convexity for scalability. In *ICML 2006*, pages 201–208. ACM, 2006.

[3] C. B. Do, S. S. Gross, and S. Batzoglou. CONTRAlign: discriminative training for protein sequence alignment. In *RECOMB*, pages 160–174, 2006.

[4] C. B. Do, D. A. Woods, and S. Batzoglou. CONTRAfold: RNA secondary structure prediction without physics-based models. *Bioinformatics*, 22(14):e90–e98, 2006.

[5] S. R. Eddy. Non-coding RNA genes and the modern RNA world. *Nature Reviews Genetics*, 2(12):919–929, 2001.

[6] Y. Freund, R. Iyer, R.E. Schapire, and Y. Singer. An efficient boosting algorithm for combining preferences. In *ICML 1998*, pages 170–178., 1998.

[7] S. Griffiths-Jones, S. Moxon, M. Marshall, A. Khanna, S. R. Eddy, and A. Bateman. Rfam: annotating non-coding RNAs in complete genomes. *Nucl. Acids Res.*, 33:D121–D124, 2005.

[8] R. Herbrich, T. Graepel, and K. Obermayer. Large margin rank boundaries for ordinal regression. In *Advances in Large Margin Classifiers*, pages 115–132, 2000. MIT Press.

[9] T. Hoang. DC optimization: Theory, methods, and applications. In R. Horst and P. Pardalos, editors, *Handbook of Global Optimization*, Kluwer.

[10] T. Joachims. Optimizing search engines using clickthrough data. In *KDD*. ACM, 2002.

[11] T. Joachims, T. Galor, and R. Elber. Learning to align sequences: A maximum-margin approach. In *New Algorithms for Macromolecular Simulation*, *LNCS 49*, 57–68. Springer, 2005.

[12] Q. Le and A.J. Smola. Direct optimization of ranking measures. NICTA-TR, 2007.

[13] T.-Y. Liu, J. Xu, T. Qin, W. Xiong, and H. Li. Letor: Benchmark dataset for research on learning to rank for information retrieval. In *LR4IR*, 2007.

[14] J. Pei and N. V. Grishin. MUMMALS: multiple sequence alignment improved by using hidden Markov models with local structural information. *Nucl. Acids Res.*, 34(16):4364–4374, 2006.

[15] N. Ratliff, J. Bagnell, and M. Zinkevich. (online) subgradient methods for structured prediction. In *AISTATS*, 2007.

[16] B. Rost. Twilight zone of protein sequence alignments. *Protein Eng.*, 12(2):85–94, 1999.

[17] S. Shalev-Shwartz, Y. Singer, and N. Srebro. Pegasos: Primal estimated sub-gradient solver for svm. In *Proc. Intl. Conf. Machine Learning*, 2007.

[18] B. Taskar, C. Guestrin, and D. Koller. Max-margin Markov networks. In *NIPS 16*, pages 25–32, 2004. MIT Press.

[19] C.H. Teo, Q. Le, A.J. Smola, and S.V.N. Vishwanathan. A scalable modular convex solver for regularized risk minimization. In *KDD*. ACM, 2007.

[20] I. Tsochantaridis, T. Joachims, T. Hofmann, and Y. Altun. Large margin methods for structured and interdependent output variables. *J. Mach. Learn. Res.*, 6:1453–1484, 2005.

[21] M. Weimer, A. Karatzoglou, Q. Le, and A. Smola. Cofi rank - maximum margin matrix factorization for collaborative ranking. In *NIPS 20*. MIT Press, 2008.

[22] A.L. Yuille and A. Rangarajan. The concave-convex procedure. *Neural Computation*, 15:915–936, 2003.

[23] A. Nedic and D. P. Bertsekas. Incremental subgradient methods for nondifferentiable optimization. *Siam J. on Optimization*, 12:109–138, 2001.

